# A Computational Basis for Phonology

**David S. Touretzky**
School of Computer Science
Carnegie Mellon University
Pittsburgh, PA 15213

**Deirdre W. Wheeler**
Department of Linguistics
University of Pittsburgh
Pittsburgh, PA 15260

## ABSTRACT

The phonological structure of human languages is intricate, yet highly constrained. Through a combination of connectionist modeling and linguistic analysis, we are attempting to develop a computational basis for the nature of phonology. We present a connectionist architecture that performs multiple simultaneous insertion, deletion, and mutation operations on sequences of phonemes, and introduce a novel additional primitive, *clustering*. Clustering provides an interesting alternative to both iterative and relaxation accounts of assimilation processes such as vowel harmony. Our resulting model is efficient because it processes utterances entirely in parallel using only feed-forward circuitry.

## 1   INTRODUCTION

Phonological phenomena can be quite complex, but human phonological behavior is also highly constrained. Many operations that are easily learned by a perceptron-like sequence mapping network are excluded from real languages. For example, as Pinker and Prince (1988) point out in their critique of the Rumelhart and McClelland (1986) verb learning model, human languages never reverse the sequence of segments in a word, but this is an easy mapping for a network to learn. On the other hand, we note that some phonological processes that are relatively common in human languages, such as vowel harmony, appear difficult for a sequence-mapping architecture to learn. Why are only certain types of sequence operations found in human languages, and not others? We suggest that this is a reflection of the limitations of an underlying, genetically-determined, specialized computing architecture. We are searching for this architecture.

Our work was initially inspired by George Lakoff's theory of cognitive phonology (Lakoff, 1988, 1989), which is in turn a development of the ideas of John Goldsmith (to appear). Lakoff proposes a three-level representation scheme. The M (morphophonemic) level represents the underlying form of an utterance, the P (phonemic) level is an intermediate form, and the F (phonetic) level is the derived surface form.

Lakoff uses a combination of inter-level mapping rules and intra-level well-formedness conditions to specify the relationships between P- and F-level representations and the M-level input. In a connectionist implementation, the computations performed by the mapping rules are straightforward, but we find the well-formedness conditions troubling. Goldsmith's proposal was that phonology is a goal-directed constraint satisfaction system that operates via parallel relaxation. He cites Smolensky's harmony theory[1] Lakoff has adopted this appeal to harmony theory in his description of how well-formedness conditions could work.

In our model, we further develop the Goldmsith and Lakoff mapping scheme, but we reject harmony-based well-formedness conditions for several reasons. First, harmony theory involves simulated annealing search. The timing constraints of real nervous systems rule out simulated annealing. Second, it is not clear how to construct an energy function for a connectionist network that performs complex discrete phonological operations. Finally there is our desire to explain why certain types of processes occur in human languages and others do not. Harmony theory alone is too unconstrained for this purpose.

We have implemented a model called $M^3P$ (for "Many Maps" Model of Phonology) that allows us to account for virtually all of the phenomena in (Lakoff, 1989) using a tightly-constrained, purely-feedforward computing scheme. In the next section we describe the mapping matrix architecture that is the heart of $M^3P$. Next we give an example of an iterative process, Yawelmani vowel harmony,[2] which Lakoff models with a P-level well-formedness condition. Such a condition would have to be implemented by relaxation search for a "minimum energy state" in the P-level representation, which we wish to avoid. Finally we present our alternative approach to vowel harmony, using a novel clustering mechanism that eliminates the need for relaxation.

## 2    THE MAPPING MATRIX ARCHITECTURE

Figure 1 is an overview of our "many maps" model. M-P constructions compute how to go from the M-level representation of an utterance to the P-level representation. The derivation is described as a set of explicit changes to the M-level string. M-P constructions read the segments in the M-level buffer and write the changes, phrased as mutation, deletion, and insertion requests, into slots of a buffer called P-deriv. The M-level and P-deriv buffers are then read by the M-P mapping matrix, which produces the P-level representation as its output. The process is repeated at the next level, with P-F constructions writing changes into an F-deriv buffer, and a P-F map deriving an F-level

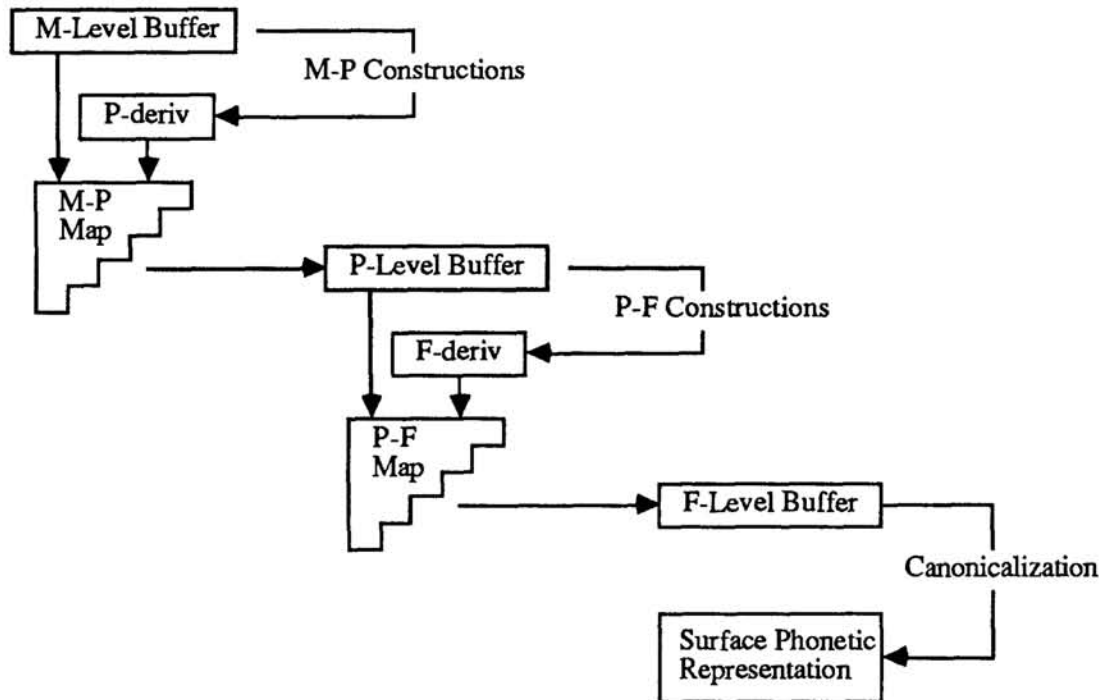

**Figure 1:** Overview of the "many maps" model.

representation. A final step called "canonicalization" cleans up the representations of the individual segments.

Figure 2 shows the effect of an M-P construction that breaks up CCC consonant clusters by inserting a vowel after the first consonant, producing CiCC. The input in this case is the Yawelmani word /?ugnhin/ "drinks", and the desired insertion is indicated in P-deriv. The mapping matrix derives the P-level representation right-justified in the buffer, with no segment gaps or collisions. It can do this even when mutliple simultaneous insertions and deletions are being performed. But it cannot perform arbitrary sequence manipulations, such as reversing all the segments of an utterance. Further details of the matrix architecture are given in (Touretzky, 1989) and (Wheeler and Touretzky, 1989).

## 3   ITERATIVE PHENOMENA

Several types of phonological processes operate on groups of adjacent segments, often by making them more similar to an immediately preceding (or following) trigger segment. Vowel harmony and voicing assimilation are two examples. In Yawelmani, vowel harmony takes the following form: an [αhigh] vowel that is preceded by an [αhigh] round vowel becomes round and back. In the form /do:s+al/ "might report", the non-round, back vowel /a/ is [−high], as is the preceding round vowel /o/. Therefore the /a/ becomes round, yielding the surface form [do:sol]. Similarly, in /dub+hin/ "leads by the hand", the [+high] vowel /i/ is preceded by the [+high] round vowel /u/, so the /i/ becomes round and back, giving [dubhun]. In /bok'+hin/ "finds", the /i/ does not undergo harmony because it differs in height from the preceding vowel.

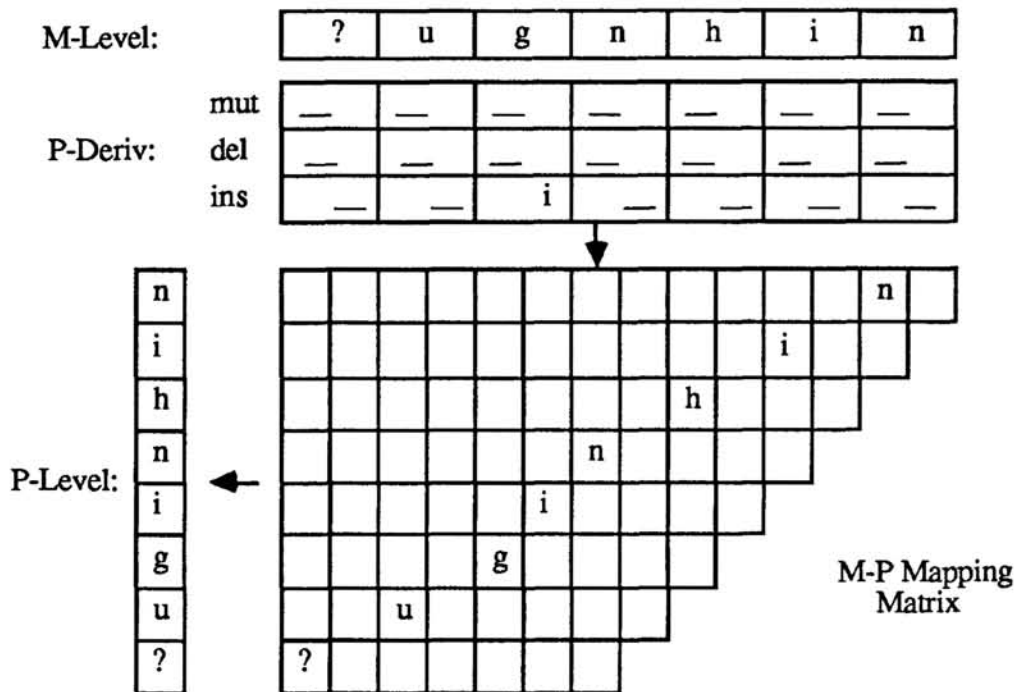

**Figure 2:** Performing an insertion via the M-P mapping matrix.

Harmony is described as an iterative process because it can apply to entire sequences of vowels, as in the following derivation:

| /t'ul+sit+hin/ | "burns for" |
| /t'ul+sut+hin/ | *harmony on second vowel* |
| /t'ul+sut+hun/ | *harmony on third vowel* |

In Yawelmani we saw an epenthesis process that inserts a high vowel /i/ to break up lengthy consonant clusters. Epenthetic vowels may either undergo or block harmony. With the word /logw+xa/ "let's pulverize", epenthesis inserts an /i/ to break up the /gwx/ cluster, producing /logiw+xa/. Now the /a/ is preceded by a [+high, −round] vowel, so harmony does not apply, whereas in /do:s+al/, which has the same sequence of underlying vowels, it did. This is an instance of epenthesis blocking harmony. In other environments the epenthetic vowel may itself undergo harmony. For example:

| /ʔugn+hin/ | "drinks" |
| /ʔuginhin/ | *epenthesis* |
| /ʔugunhin/ | *harmony on epenthetic vowel* |
| /ʔugunhun/ | *harmony on third vowel* |

The standard generative phonology analysis of harmony utilizes the following rule, applying after epenthesis, that is supposed to iterate through the utterance from left to right, changing one vowel at a time:

$$\begin{bmatrix} +syll \\ \alpha high \end{bmatrix} \rightarrow \begin{bmatrix} +round \\ +back \end{bmatrix} / \begin{bmatrix} +syll \\ +round \\ \alpha high \end{bmatrix} C_0 \underline{\hphantom{xx}}$$

Lakoff offers an alternative account of epenthesis and harmony that eliminates iteration. He states epenthesis as an M-P construction:

```
M:      C        C    {C,#}
        |    |   |
P:      []   i   []
```

The harmony rule is stated as a P-level well-formedness condition that applies simultaneously throughout the buffer:

> P:      If [+syll, +round, $\alpha$high] $C_0$ X,
>            then if X = [+syll, $\alpha$high], then X = [+round, +back].

Starting with /ʔugn+hin/ at M-level, Lakoff's model would settle into a representation of /ʔugunhun/ at P-level. We repeat again the crucial point that this representation is not derived by sequential application of rules; it is merely *licensed* by one application of epenthesis and two of harmony. The actual computation of the P-level representation would be performed by a parallel relaxation process, perhaps using simulated annealing, that somehow determines the sequence that best satisfies all applicable constraints at P-level.

## 4   THE CLUSTERING MECHANISM

Our account of vowel harmony must differ from Lakoff's because we do not wish to rely on relaxation in our model. Instead, we introduce special clustering circuitry to recognize sequences of segments that share certain properties. The clustering idea is meant to be analogous to perceptual grouping in vision. Sequences of adjacent visually-similar objects are naturally perceived as a whole. A similar mechanism operating on phonological sequences, although unprecedented in linguistic theory, does not appear implausible. Crucial to our model is the principle that perceived sequences may be operated on as a unit. This allows us to avoid iteration and give a fully-parallel account of vowel harmony.

The clustering mechanism is controlled by a small number of language-specific parameters. The rule shown below is the P-F clustering rule for Yawelmani. Cluster type [+syllabic] indicates that the rule looks only at vowels. (This is implemented by an additional mapping matrix that extracts the vowel projection of the P-level buffer. The clustering mechanism actually looks at the output of this matrix rather than at the P-level buffer directly.) The trigger of a cluster is a round vowel of a given height, and the elements are the subsequent adjacent vowels of matching height. Application of the rule causes elements (but not triggers) to undergo a change; in this case, they become round and back.

Yawelmani vowel harmony — P-F mapping:
Cluster type:   [+syllabic]
Trigger:        [+round, $\alpha$high]
Element:        [$\alpha$high]
Change:         [+round, +back]

The following hypothetical vowel sequence illustrates the application of this clustering rule. Consonants are omitted for clarity:

|  | 1 | 2 | 3 | 4 | 5 | 6 | 7 | 8 | 9 |
|---|---|---|---|---|---|---|---|---|---|
|  | i | u | i | i | e | o | o | a | i |
| trigger: |  | + |  |  |  | + |  |  |  |
| element: |  |  | + | + |  |  | + | + |  |

The second vowel is round, so it's a trigger. Since the third and fourth vowels match it in height, they become elements. The fifth vowel is [−high], so it is not included in the cluster. The sixth vowel triggers a new cluster because it's round; it is also [−high]. The seventh and eighth vowels are also [−high], so they can be elements, but the ninth vowel is excluded from the cluster because is [+high]. Note that vowel 7 is an element, but it also meets the specification for a trigger. Given a choice, our model prefers to mark segments as elements rather than triggers because only elements undergo the specified change. The distinction is moot in Yawlemani, where triggers are already round and back, but it matters in other languages; see (Wheeler and Touretzky, 1989) for details.

Figures 2 and 3 together show the derivation of the Yawelmani word [ʔugunhun] from the underlying form /ʔugn+hin/. In figure 2 an M-P construction inserted a high vowel. In figure 3 the P-F clustering circuitry has examined the P-level buffer and marked the triggers and elements. Segments that were marked as elements then have the change [+round, +back] written into their corresponding mutation slots in F-deriv. Finally, the P-F mapping matrix produces the sequence /ʔugunhun/ as the F-level representation of the utterance.

## 5   DISCUSSION

We could not justify the extra circuitry required for clustering if it were suitable only for Yawelmani vowel harmony. The same mechanism handles a variety of other iterative phenomena, including Slovak and Gidabal vowel shortening, Icelandic umlaut, and Russian voicing assimilation. The full mechanism has some additional parameters beyond those covered in the discussion of Yawelmani. For example, clustering may proceed from right-to-left (as is the case in Russian) instead of from left-to-right. Also, clusters may be of either bounded or unbounded length. Bounded clusters are required for alternation processes, such as Gidabal shortening. They cover exactly two segments: a trigger and one element. We are making a deliberate analogy here with metrical phonology (stress systems), where unbounded feet may be of arbitrary length, but bounded feet always contain exactly two syllables. No language has strictly trisyllabic feet. We predict a similar constraint will hold for iterative phenomena when they are reformulated in parallel clustering terms, i.e., no language requires bounded-length clusters with more than one element.

P-level:

| ? | u | g | i | n | h | i | n |

Clustering:

trigger
element

| + |  |  |  |  |  |  |
|  |  | + |  |  |  | + |

F-deriv:

mut
del
ins

| — | — | — | +rnd | — | — | +rnd | — |
| — | — | — | — | — | — | — | — |
| — | — | — | — | — | — | — | — |

F-level:

| n |
| u |
| h |
| n |
| ü |
| g |
| u |
| ? |

P-F Mapping
Matrix

**Figure 3:** Clustering applied to Yawelmani vowel harmony.

Our model makes many other predictions of constraints on human phonology, based on limitations of the highly-structured "many maps" architecture. We are attempting to verify these predictions, and also to extend the model to additional aspects of phonological behavior, such as syllabification and stress.

## Acknowledgements

This research was supported by a contract from Hughes Research Laboratories, by the Office of Naval Research under contract number N00014-86-K-0678, and by National Science Foundation grant EET-8716324. We thank George Lakoff for encouragement and support, John Goldsmith for helpful correspondence, and Gillette Elvgren III for implementing the simulations.

## Footnotes

[1] Smolensky's "harmony theory" should not be confused with the linguistic phenomenon of "vowel harmony."

[2] Yawelmani is a dialect of Yokuts, an American Indian language from California. Our Yawelmani data is drawn from Kenstowicz and Kisseberth (1979), as is Lakoff's.

## References

Goldsmith, J. (to appear) Phonology as an intelligent system. To appear in a festschrift for Leila Gleitman, edited by D. Napoli and J. Kegl.

Kenstowicz, M., and Kisseberth, C. (1979) *Generative Phonology: Description and Theory*. San Diego, CA: Academic Press.

Lakoff, G. (1988) A suggestion for a linguistics with connectionist foundations. In D. S. Touretzky, G. E. Hinton, and T. J. Sejnowski (eds.), *Proceedings of the 1988 Connectionist Models Summer School*, pp. 301-314. San Mateo, CA: Morgan Kaufmann.

Lakoff, G. (1989) Cognitive phonology. Draft of paper presented at the UC-Berkeley Workshop on Constraints vs Rules, May 1989.

Pinker, S., and Prince, A. (1988) On language and connectionism: analysis of a parallel distributed processing model of language acquisition. In S. Pinker & J. Mehler (eds.), *Connections and Symbols*. Cambridge, Massachusetts: MIT Press.

Rumelhart, D. E., and McClelland, J. L. (1986) On learning the past tenses of English verbs. In J. L. McClelland and D. E. Rumelhart (eds.), *Parallel Distributed Processing: Explorations in the MicroStructure of Cognition*, volume 2. Cambridge, Massachusetts: MIT Press.

Smolensky, P. (1986) Information processing in dynamical systems: foundations of harmony theory. In D. E. Rumelhart and J. L. McClelland (eds.), *Parallel Distributed Processing: Explorations in the MicroStructure of Cognition*, volume 1. Cambridge, Massachusetts: MIT Press.

Touretzky, D. S. (1989) Toward a connectionist phonology: the "many maps" approach to sequence manipulation. *Proceedings of the Eleventh Annual Conference of the Cognitive Science Society*, pp. 188-195. Hillsdale, NJ: Erlbaum.

Wheeler, D. W., and Touretzky, D. S. (1989) A connectionist implementation of cognitive phonology. Technical report CMU-CS-89-144, Carnegie Mellon University, School of Computer Science. To appear in G. Lakoff and L. Hyman (eds.), *Proceedings of the UC-Berkeley Phonology Workshop on Constraints vs, Rules*, University of Chicago Press.